# Active dendrites:
# adaptation to spike-based communication

**Balázs B Ujfalussy**[1,2]
ubi@rmki.kfki.hu

**Máté Lengyel**[1]
m.lengyel@eng.cam.ac.uk

[1] Computational & Biological Learning Lab, Dept. of Engineering, University of Cambridge, UK
[2] Computational Neuroscience Group, Dept. of Biophysics, MTA KFKI RMKI, Budapest, Hungary

## Abstract

Computational analyses of dendritic computations often assume stationary inputs to neurons, ignoring the pulsatile nature of spike-based communication between neurons and the moment-to-moment fluctuations caused by such spiking inputs. Conversely, circuit computations with spiking neurons are usually formalized without regard to the rich nonlinear nature of dendritic processing. Here we address the computational challenge faced by neurons that compute and represent analogue quantities but communicate with digital spikes, and show that reliable computation of even purely linear functions of inputs can require the interplay of strongly nonlinear subunits within the postsynaptic dendritic tree. Our theory predicts a matching of dendritic nonlinearities and synaptic weight distributions to the joint statistics of presynaptic inputs. This approach suggests normative roles for some puzzling forms of nonlinear dendritic dynamics and plasticity.

## 1   Introduction

The operation of neural circuits fundamentally depends on the capacity of neurons to perform complex, nonlinear mappings from their inputs to their outputs. Since the vast majority of synaptic inputs impinge the dendritic membrane, its morphology, and passive as well as active electrical properties play important roles in determining the functional capabilities of a neuron. Indeed, both theoretical and experimental studies suggest that active, nonlinear processing in dendritic trees can significantly enhance the repertoire of singe neuron operations [1, 2].

However, previous functional approaches to dendritic processing were limited because they studied dendritic computations in a firing rate-based framework [3, 4], essentially requiring both the inputs and the output of a cell to have stationary firing rates for hundreds of milliseconds. Thus, they ignored the effects and consequences of temporal variations in neural activities at the time scale of inter-spike intervals characteristic of *in vivo* states [5]. Conversely, studies of spiking network dynamics [6, 7] have ignored the complex and highly nonlinear effects of the dendritic tree.

Here we develop a computational theory that aims at explaining some of the morphological and electrophysiological properties of dendritic trees as adaptations towards spike-based communication. In line with the vast majority of theories about neural network computations, the starting point of our theory is that each neuron needs to compute some function of the membrane potential (or, equivalently, the instantaneous firing rate) of its presynaptic partners. However, as the postsynaptic neuron does not have direct access to the presynaptic membrane potentials, only to the spikes emitted by its presynaptic partners based on those potentials, computing the required function becomes a non-trivial inference problem. That is, neurons need to perform computations on their inputs in the face of significant uncertainty as to what those inputs exactly are, and so as to what their required output might be.

In section 2 we formalize the problem of inferring some required output based on incomplete spiking-based information about inputs and derive an optimal online estimator for some simple

but tractable cases. In section 3 we show that the optimal estimator exhibits highly nonlinear behavior closely matching aspects of active dendritic processing, even when the function of inputs to be computed is purely linear. We also present predictions about how the statistics of presynaptic inputs should be matched by the clustering patterns of synaptic inputs onto active subunits of the dendritic tree. In section 4 we discuss our findings and ways to test our predictions experimentally.

## 2 Estimation from correlated spike trains

### 2.1 The need for nonlinear dendritic operations

Ideally, the (subthreshold) dynamics of the somatic membrane potential, $v(t)$, should implement some nonlinear function, $f(\mathbf{u}(t))$, of the presynaptic membrane potentials, $\mathbf{u}(t)$.[1]

$$\tau \frac{\mathrm{d}v(t)}{\mathrm{d}t} = f(\mathbf{u}(t)) - v(t) \tag{1}$$

However, the presynaptic membrane potentials cannot be observed directly, only the presynaptic spike trains $\mathbf{s}_{0:t}$ that are stochastic functions of the presynaptic membrane potential trajectories. Therefore, to minimise squared error, the postsynaptic membrane potential should represent the mean of the posterior over possible output function values it should be computing based on the input spike trains:

$$\tau \frac{\mathrm{d}v(t)}{\mathrm{d}t} \simeq \int f(\mathbf{u}(t)) \; \mathrm{P}(\mathbf{u}(t)|\mathbf{s}_{0:t}) \; \mathrm{d}\mathbf{u}(t) - v(t) \tag{2}$$

Biophysically, to a first approximation, the somatic membrane potential of the postsynaptic neuron can be described as some function(al), $\tilde{f}$, of the local dendritic membrane potentials, $\mathbf{v}^{\mathrm{d}}(t)$

$$\tau \frac{\mathrm{d}v(t)}{\mathrm{d}t} = \tilde{f}(\mathbf{v}^{\mathrm{d}}(t)) - v(t) \tag{3}$$

This is interesting because Pfister et al. [11, 12] have recently suggested that short-term synaptic plasticity arranges for each local dendritic postsynaptic potential, $v_i^{\mathrm{d}}$, to (approximately) represent the posterior mean of the corresponding presynaptic membrane potential:

$$v_i^{\mathrm{d}}(t) \simeq \int u_i(t) \, \mathrm{P}(u_i(t)|s_{i,0:t}) \; \mathrm{d}u_i \tag{4}$$

Thus, it would be tempting to say that in order to achieve the computational goal of Eq. 2, the way the dendritic tree (together with the soma) should integrate these local potentials, as given by $\tilde{f}$, should be directly determined by the function that needs to be computed: $\tilde{f} = f$. However, it is easy to see that in general this is going to be incorrect:

$$f\left(\int \mathbf{u}(t) \prod_i \mathrm{P}(u_i(t)|s_{i,0:t}) \; \mathrm{d}\mathbf{u}(t)\right) \neq \int f(\mathbf{u}(t)) \; \mathrm{P}(\mathbf{u}(t)|\mathbf{s}_{0:t}) \; \mathrm{d}\mathbf{u}(t) \tag{5}$$

where the l.h.s. is what the neuron implements (eqs. 3-4) and the r.h.s. is what it should compute (eq. 2). The equality does not hold in general when $f$ is non-linear or $\mathrm{P}(\mathbf{u}(t)|\mathbf{s}_{0:t})$ does not factorise.

In the following, we are going to consider the case when the function, $f(\mathbf{u})$, is a purely linear combination of synaptic inputs, $f(\mathbf{u}) = \sum_i c_i \, u_i$. Such linear transformations seem to suggest linear dendritic operations and, in combination with a single global 'somatic' nonlinearity, they are often assumed in neural network models and descriptive models of neuronal signal processing [10]. However, as we will show below, estimation from the spike trains of multiple correlated presynaptic neurons requires a non-linear integration of inputs even in this case.

## 2.2 The mOU-NP model

We assume that the hidden dynamics of presynaptic membrane potentials are described by a multi-variate Ornstein–Uhlenbeck (mOU) process (discretised in time into $\delta t \to 0$ time bins, thus formally yielding an AR(1) process):

$$\mathbf{u}_t = \mathbf{u}_{t-\delta t} + \frac{1}{\tau}(u_0 - \mathbf{u}_{t-\delta t})\delta t + \mathbf{q}_t\sqrt{\delta t}, \qquad \mathbf{q}_t \overset{\text{iid}}{\sim} \mathcal{N}(\mathbf{0}, \mathbf{Q}) \tag{6}$$

$$= \alpha\mathbf{u}_{t-\delta t} + \mathbf{q}_t\sqrt{\delta t} + \frac{\delta t}{\tau}u_0 \tag{7}$$

where we described all neurons with the same parameters: $u_0$, the resting potential and $\tau$, the membrane time constant (with $\alpha = 1 - \frac{\delta t}{\tau}$). Importantly, $\mathbf{Q}$ is the covariance matrix parametrising the correlations between the subthreshold membrane potential fluctuations of presynaptic neurons.

Spiking is described by a nonlinear-Poisson (NP) process where the instantaneous firing rate, $r$, is an exponential function of $u$ with exponent $\beta$ and "baseline rate" $g$:

$$r(u) = g\,e^{\beta u} \tag{8}$$

and the number of spikes emitted in a time bin, $s$, is Poisson with this rate:

$$P(s|u) = \text{Poisson}(s; \delta t\,r(u)) \tag{9}$$

The spiking process itself is independent i.e., the likelihood is factorised across cells:

$$P(\mathbf{s}|\mathbf{u}) = \prod_i P(s_i|u_i) \tag{10}$$

## 2.3 Assumed density filtering in the mOU-NP model

Our goal is to derive the time evolution of the posterior distribution of the membrane potential, $P(\mathbf{u}_t|\mathbf{s}_{0:t})$, given a particular spiking pattern observed. Ultimately, we will need to compute some function of $\mathbf{u}$ under this distribution. For linear computations (see above), the final quantity of interest depends on $\sum_i c_i\,u_i$ which in the limit (of many presynaptic cells) is going to be Gaussian-distributed, and as such only dependent on the first two moments of the posterior. This motivates us to perform assumed density filtering, by which we substitute the true posterior with a moment-matched multivariate Gaussian in each time step, $P(\mathbf{u}_t|\mathbf{s}_{0:t}) \simeq \mathcal{N}(\mathbf{u}_t; \boldsymbol{\mu}_t, \boldsymbol{\Sigma}_t)$.

After some algebra (see Appendix for details) we obtain the following equations for the time evolution of the mean and covariance of the posterior under the generative process defined by Eqs. 7-10:

$$\dot{\boldsymbol{\mu}} = \frac{1}{\tau}(u_0 - \boldsymbol{\mu}) + \beta\boldsymbol{\Sigma}(\mathbf{s}(t) - \boldsymbol{\gamma}) \tag{11}$$

$$\dot{\boldsymbol{\Sigma}} = \frac{2}{\tau}(\boldsymbol{\Sigma}_{OU} - \boldsymbol{\Sigma}) - \beta^2\boldsymbol{\Sigma}\boldsymbol{\Gamma}\boldsymbol{\Sigma} \tag{12}$$

where $s_i(t)$ is the spike train of presynaptic neuron $i$ represented as a sum of Dirac-delta functions, $\boldsymbol{\gamma}$ ($\boldsymbol{\Gamma}$) is a vector (diagonal matrix) whose elements $\gamma_i = \Gamma_{ii} = g\,e^{\beta\mu_i + \frac{\beta^2\Sigma_{ii}}{2}}$ are the estimated firing rates of the neurons, and $\boldsymbol{\Sigma}_{OU} = \frac{\mathbf{Q}\tau}{2}$ is the prior covariance matrix of the presynaptic membrane potentials in the absence of any observation.

## 2.4 Modelling correlated *up* and *down* states

The mOU-NP process is a convenient and analytically tractable way to model correlations between presynaptic neurons but it obviously falls short of the dynamical complexity of cortical ensembles in many respects. Following and expanding on [12], here we considered one extension that allowed us to model coordinated changes between more hyper- and depolarised states across presynaptic neurons, such as those brought about by cortical *up* and *down* states.

In this extension, the 'resting' potential of each presynaptic neuron, $u_0$, could switch between two different values, $u_{\text{up}}$ and $u_{\text{down}}$, and followed first order Markovian dynamics. *Up* and *down* states in cortical neurons are not independent but occur synchronously [13]. To reproduce these correlations

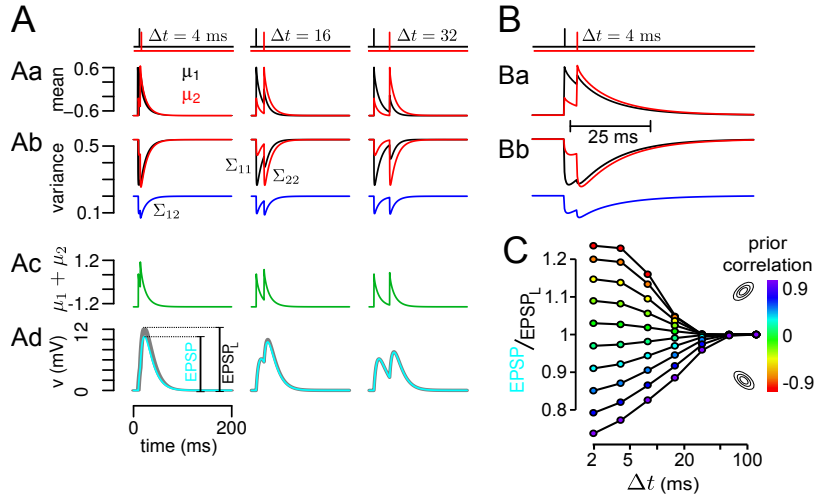

Figure 1: Simulation of the optimal estimator in the case of two presynaptic spikes with different time delays ($\Delta t$). A: The posterior means (Aa), variances, $\Sigma_{ii}$, and the covariance, $\Sigma_{12}$ (Ab). The dynamics of the postsynaptic membrane potential, $v$ (Ad) is described by Eq. 1, where $f(\mathbf{u}) = u_1 + u_2$ (Ac). B: The same as A on an extended time scale. C: The nonlinear summation of two EPSPs, characterised by the ratio of the actual EPSP (cyan on Ad) and the linear sum of two individual EPSPs (grey on Ad) is shown for different delays and correlations between the presynaptic neurons. The summation is sublinear if the presynaptic neurons are positively correlated, whereas negative correlations imply supralinear summation.

we introduced a global, binary state variable, $x$ that influenced the Markovian dynamics of the resting potential of individual neurons (see Appendix and Fig. 2A). Unfortunately, an analytical solution to the optimal estimator was out of reach in this case, so we resorted to particle filtering [14] to compute the output of the optimal estimator.

## 3 Nonlinear dendrites as near-optimal estimators

### 3.1 Correlated Ornstein-Uhlenbeck process

First, we analysed the estimation problem in case of mOU dynamics where we could derive an optimal estimator for the membrane potential. Postsynaptic dynamics needed to follow the linear sum of presynaptic membrane potentials. Figure 1 shows the optimal postsynaptic response (Eqs. 11-12) after observing a pair of spikes from two correlated presynaptic neurons with different time delays. When one of the cells (black) emits a spike, this causes an instantaneous increase not only in the membrane potential estimate of the neuron itself but also in those of all correlated neurons (red neuron in Fig. 1Aa and Ba). Consequently, the estimated firing rate, $\gamma$, of both cells increases. Albeit indirectly, a spike also influences the uncertainty about the presynaptic membrane potentials – quantified by the posterior covariance matrix. A spike itself does not change this covariance directly, but since it increases estimated firing rates, the absence of even more spikes in the subsequent period becomes more informative. This increased information rate following a spike decreases estimator uncertainty about true membrane potential values for a short period (Fig. 1Ab and Bb). However, as the estimated firing rate decreases back to its resting value nearly exponentially after the spike, the estimated uncertainty also returns back to its steady state.

Importantly, the instantaneous increase of the posterior means in response to a spike is proportional to the estimated uncertainty about the membrane potentials and to the estimator's current belief about the correlations between the neurons. As each spike influences not only the mean estimate of the membrane potentials of other correlated neurons but also the uncertainty of these estimates, the effect of a spike from one cell on the posterior mean depends on the spiking history of all other correlated neurons (Fig. 1Ac-Ad).

In the example shown in Fig. 1, the postsynaptic dynamics is required to compute a purely linear sum of two presynaptic membrane potentials, $f(\mathbf{u}) = u_1 + u_2$. However, depending on the prior correlation between the two presynaptic neurons and the time delay between the two spikes, the amplitude of the postsynaptic membrane potential change evoked by the pair of spikes can be either larger or smaller than the linear sum of the individual excitatory postsynaptic potentials (EPSPs) (Fig. 1Ad, C). EPSPs from independent neurons are additive, but if the presynaptic neurons are positively correlated then their spikes convey redundant information and they are integrated sublinearly. Conversely, simultaneous spikes from negatively correlated presynaptic neurons are largely unexpected and induce supralinear summation. The deviation from the linear summation is proportional to the magnitude of the correlation between the presynaptic neurons (Fig. 1C).

We compared the nonlinear integration of the inputs in the optimal estimator with experiments measuring synaptic integration in the dendritic tree of neurons. For a passive membrane, cable theory [15] implies that inputs are integrated linearly only if they are on electronically separated dendritic branches, but reduction of the driving force entails a sublinear interaction between co-localised inputs. Moreover, it has been found that active currents, the $I_A$ potassium current in particular, also contribute to the sublinear integration within the dendritic tree [16, 17]. Our model predicts that inputs that are integrated sublinearly are positively correlated (Fig. 1C).

In sum, we can already see that correlated inputs imply nonlinear integration in the postsynaptic neuron, and that the form of nonlinearity needs to be matched to the degree and sign of correlations between inputs. However, the finding that supralinear interactions are only expected from *anti*correlated inputs defeats biological intuition. Another shortcoming of the mOU model is related to the second-order effects of spikes on the posterior covariance. As the covariance matrix does not change instantaneously after observing a presynaptic spikes (Fig. 1B), two spikes arriving simultaneously are summed linearly (not shown). At the other extreme, two spikes separated by long delays again do not influence each other. Therefore the nonlinearity of the integration of two spikes has a non-monotonic shape, which again is unlike the monotonic dependence of the degree of nonlinearity on interspike intervals found in experiments [18, 19]. In order to overcome these limitations, we extended the model to incorporate correlated changes in the activity levels of presynaptic neurons [13].

### 3.2 Correlated *up* and *down* states

While the statistics of presynaptic membrane potentials exhibit more complex temporal dependencies in the extended model (Fig. 2A), importantly, the task is still assumed to be the same simple linear computation as before: $f(\mathbf{u}) = u_1 + u_2$.

However, the more complex $P(\mathbf{u})$ distribution means that we need to sum over the possible values of the hidden variables: $P(\mathbf{u}) = \sum_{\mathbf{u}_0} P(\mathbf{u}|\mathbf{u}_0) P(\mathbf{u}_0)$. The observation of a spike changes both the conditional distributions, $P(\mathbf{u}|\mathbf{u}_0)$, and the probability of being in the *up* state, $P(\mathbf{u}_0 = u_{\text{up}})$, by causing an upward shift in both. A second spike causes a further increase in the membrane potential estimate, and, more importantly, in the probability of being in the *up* state for both neurons. Since the probability of leaving the *up* state is low, the membrane potential estimate decays back to its steady state more slowly if the probability of being in the *up* state is high (Fig. 2B). This causes a supralinear increase in the membrane potential of the postsynaptic neuron which again depends on the interspike interval, but this time supralinearity is predicted for positively correlated presynaptic neurons (Fig. 2C,E). Note, that while in the mOU model, supralinear integration arises due to dynamical changes in uncertainty (of membrane potential estimates), in the extended model it is associated with a change in a hypothesis (about hidden up-down states).

This is qualitatively similar to what was found in pyramidal neurons in the neocortex [19] and in the hippocampus [18, 20] that are able to switch from (sub)linear to supralinear integration of synaptic inputs through the generation of dendritic spikes [21]. Specifically, in neocortical pyramidal neurons Polsky et al. [19] found, that nearly synchronous inputs arriving to the same dendritic branch evoke substantially larger postsynaptic responses than expected from the linear sum of the individual responses (Fig. 2D-E). While there is a good qualitative match between model and experiments, the time scales of integration are off by a factor of 2. Nevertheless, given that we did not perform exhaustive parameter fitting in our model, just simply set parameters to values that produced realistic presynaptic membrane potential trajectories (cf. our Fig. 2A with [13]), we regard the match acceptable and are confident that with further fine tuning of parameters the match would also improve quantitatively.

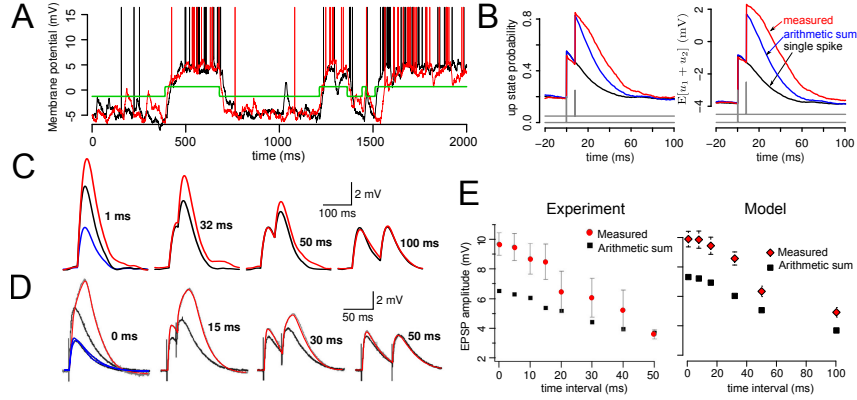

Figure 2: A: Example voltage traces and spikes from the modeled presynaptic neurons (black and red) with correlated *up* and *down* states. The green line indicates the value of the global *up-down* state variable. B: Inference in the model: The posterior probability of being in the *up* state (left) and the posterior mean of $\sum_i u_i$ after observing two spikes (grey) from different neurons with $\Delta t = 8$ ms latency. C: Supralinear summation in the switching mOU-NP model. D: Supralinear summation by dendritic spikes in a cortical pyramidal neuron. E: Peak amplitude of the response (red) and the linear sum (black squares) is shown for different delays in experiments (left) and the model (right). (D and left panel in E are reproduced from [19]).

### 3.3 Nonlinear dendritic trees are necessary for purely linear computations

In the previous sections we demonstrated that optimal inference based on correlated spike trains requires nonlinear interaction within the postsynaptic neuron, and we showed that the dynamics of the optimal estimator is qualitatively similar to the dynamics of the somatic membrane potential of a postsynaptic neuron with nonlinear dendritic processing. In this section we will build a simplified model of dendritic signal processing and compare its performance directly to several alternative models (see below) on a purely linear task, for which the neuron needs to compute the sum of presynaptic membrane potentials: $f(\mathbf{u}) = \sum_{i=1}^{10} u_i$.

We model the *dendritic* estimator as a two-layer feed-forward network of simple units (Fig. 3A) that has been proposed to closely mimic the repertoire of input-output transformations achievable by active dendritic trees [22]. In this model, synaptic inputs impinge on units in the first layer, corresponding to dendritic branches, where nonlinear integration of inputs arriving to a dendritic branch is modeled by a sigmoidal input-output function, and the outputs of dendritic branch units are in turn summed linearly in the single (somatic) unit of the second layer. We trained the model to estimate $f$ by changing the connection weights of the two layers corresponding to synaptic weights ($w_{ji}$) and branch coupling strengths ($\tilde{c}_j$, see Appendix, Fig. 3A).

We compared the performance of the dendritic estimator to four alternative models (Figure 3B):

1. The *linear estimator*, which is similar to the dendritic estimator except that the dendrites are linear.
2. The *independent estimator*, in which the individual synapses are independently optimal estimators of the corresponding presynaptic membrane potentials (Eq. 4) [11, 12], and the cell combines these estimates linearly. Note that the only difference between the independent estimator and the optimal estimator is the assumption implicit to the former that presynaptic cells are independent.
3. The *scaled independent estimator* still combines the synaptic potentials linearly, but the weights of each synapse are rescaled to partially correct for the wrong assumption of independence.
4. Finally, the *optimal estimator* is represented by the differential equations 11-12.

The performance of the different estimators were quantified by the estimation error normalized by the variance of the signal, $\frac{\langle(\sum_i u_i - \tilde{v}_{\text{estimator}})^2\rangle}{\text{var}[\sum_i u_i]}$. Figure 3C shows the estimation error of the five different models in the case of 10 uniformly correlated presynaptic neurons. If the presynaptic neurons

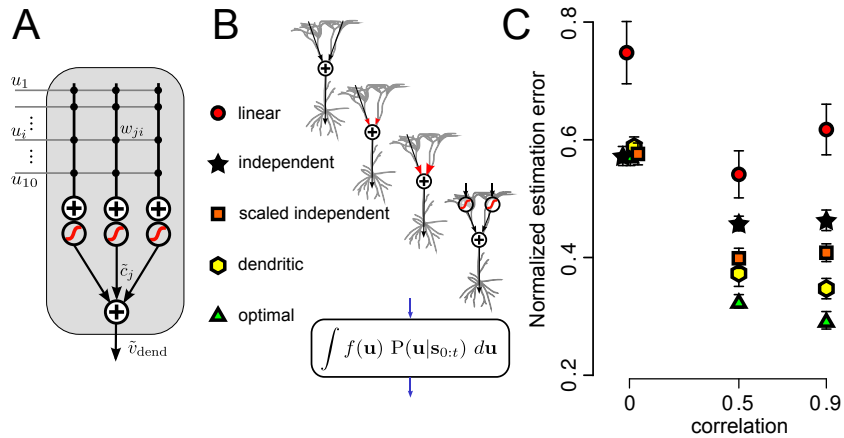

Figure 3: Performance of 5 different estimators are compared in the task of estimating $f(\mathbf{u}) = \sum_{i=1}^{N} u_i$. A: Model of the dendritic estimator. B: Different estimators (see text for more details). C: Estimation error, normalised with the variance of the signal. The number of presynaptic neurons were $N = 10$. Error bars show standard deviations.

were independent, all three estimators that used dynamical synapses ($\tilde{v}_{\mathrm{ind}}$, $\tilde{v}_{\mathrm{sind}}$ and $\tilde{v}_{\mathrm{opt}}$) were optimal, whereas the linear estimator had substantially larger error. Interestingly, the performance of the dendritic estimator (yellow) was nearly optimal even if the individual synapses were not optimal estimators for the corresponding presynaptic membrane potentials. In fact, adding depressing synapses to the dendritic model degraded its performance because the sublinear effect introduced by the saturation of the sigmoidal dendritic nonlinearity interfered with that implied by synaptic depression. When the correlation increased between the presynaptic neurons, the performance of the estimators assuming independence (black and orange) became severely suboptimal, whereas the dendritic estimator (yellow) remained closer to optimal.

Finally, in order to investigate the synaptic mechanisms underlying the remarkably high performance of the dendritic estimator, we trained a dendritic estimator on a task where the presynaptic neurons formed two groups. Neurons from different groups were independent or negatively correlated with each other, $\mathrm{cor}(u_i, u_k) = \{-0.6, -0.3, 0\}$, while there were positive correlations between neurons from the same group, $\mathrm{cor}(u_i, u_j) = \{0.3, 0.6, 0.9\}$ (Fig. 4A). The postsynaptic neuron had two dendritic branches, each of them receiving input from each presynaptic neurons initially. After tuning synaptic weights and branch coupling strengths to minimize estimation error, and pruning synapses with weights below threshold, the model achieved near-optimal performance as before (Fig. 4C). More importantly, we found that the structure of the presynaptic correlations was reflected in the synaptic connection patterns on the dendritic branches: most neurons developed stable synaptic weights only on one of the two dendritic branches, and synapses originating from neurons within the same group tended to cluster on the same branch (Fig. 4B).

## 4   Discussion

In the present paper we introduced a normative framework to describe single neuron computation that sheds new light on nonlinear dendritic information processing. Following [12], we observe that spike-based communication causes information loss in the nervous system, and neurons must infer the variables relevant for the computation [23–25]. As a consequence of this spiking bottleneck, signal processing in single neurons can be conceptually divided into two parts: the inference of the relevant variables and the computation itself. When the presynaptic neurons are independent then synapses with short term plasticity can optimally solve the inference problem [12] and nonlinear processing in the dendrites is only for computation. However, neurons in a population are often tend to be correlated [5, 13] and so the postsynaptic neuron should combine spike trains from such correlated neurons in order to find the optimal estimate of its output. We demonstrated that the solution of this inference problem requires nonlinear interaction between synaptic inputs in the

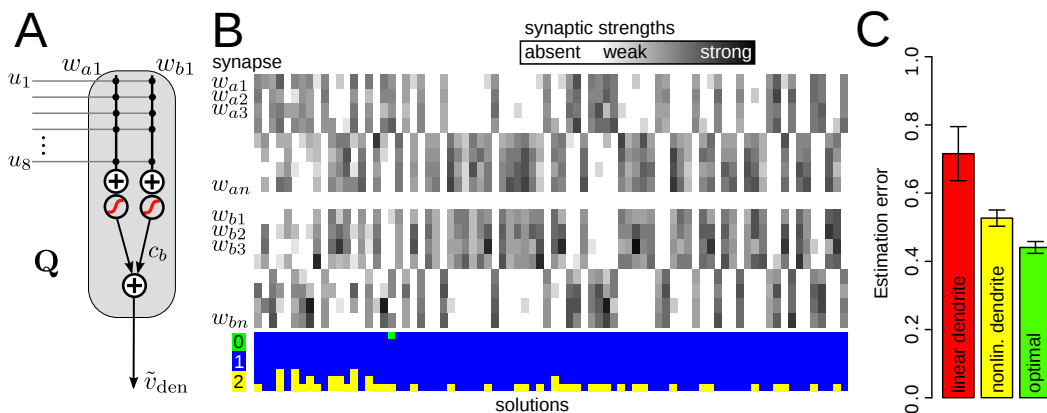

Figure 4: Synaptic connectivity reflects the correlation structure of the input. A: The presynaptic covariance matrix is block-diagonal, with two groups (neurons 1–4 and 5–8). Initially, each presynaptic neuron innervates both dendritic branches, and the weights, $w$, of the static synapses are then tuned to minimize estimation error. B: Synaptic weights after training, and pruning the weakest synapses. Columns corresponds to solutions of the error-minimization task with different presynaptic correlations and/or initial conditions, and rows are different synapses. The detailed connectivity patterns differ across solutions, but neurons from the same group usually all innervate the same dendritic branch. Below: fraction of neurons in each solution innervating 0, 1 or 2 branches. The height of the yellow (blue, green) bar indicates the proportion of presynaptic neurons innervating two (one, zero, respectively) branches of the postsynaptic neuron. C: After training, the nonlinear dendritic estimator performs close to optimal and much better than the linear neuron.

postsynaptic cell even if the computation itself is purely linear. Of course, actual neurons are usually faced with both problems: they will need to compute nonlinear functions of correlated inputs and thus their nonlinearities will serve both estimation and computation. In such cases our approach allows dissecting the respective contributions of active dendritic processing towards estimation and computation.

We demonstrated that the optimal estimator of the presynaptic membrane potentials can be closely approximated by a nonlinear dendritic tree where the connectivity from the presynaptic cells to the dendritic branches and the nonlinearities in the dendrites are tuned according to the dependency structure of the input. Our theory predicts that independent neurons will innervate distant dendritic domains, whereas neurons that have correlated membrane potentials will impinge on nearby dendritic locations, preferentially on the same dendritic branches, where synaptic integration in nonlinear [19, 26]. More specifically, the theory predicts sublinear integration between positively correlated neurons and supralinear integration through dendritic spiking between neurons with correlated changes in their activity levels. To directly test this prediction the membrane potentials of several neurons need to be recorded under naturalistic *in vivo* conditions [5, 13] and then the subcellular topography of their connectivity with a common postsynaptic target needs to be determined. Similar approaches have been used recently to characterize the connectivity between neurons with different receptive field properties *in vivo* [27, 28].

Our model suggests that the postsynaptic neuron should store information about the dependency structure of its presynaptic partners within its dendritic membrane. Online learning of this information based on the observed spiking patterns requires new, presumably non-associative forms of plasticity such as branch strength potentiation [29, 30] or activity-dependent structural plasticity [31].

### Acknowledgments

We thank J-P Pfister for valuable insights and comments on earlier versions of the manuscript, and P Dayan, B Gutkin, and Sz Káli for useful discussions. This work has been supported by the Hungarian Scientific Research Fund (OTKA, grant number: 84471, BU) and the Welcome Trust (ML).

## Footnotes

[1]Dynamics of this form are assumed by many neural network models, though the variables $\mathbf{u}$ amd $v$ are usually interpreted as instantaneous firing rates rather than membrane potentials [10]. However, just as in our case (Eq. 8), the two are often taken to be related through a simple non-linear function which thus makes the two frameworks essentially isomorphic.

# References

1. Koch, C. *Biophysics of computation* (Oxford University Press, 1999).
2. Stuart, G., Spruston, N. & Hausser, M. *Dendrites* (Oxford University Press, 2007).
3. Poirazi, P. & Mel, B.W. Impact of active dendrites and structural plasticity on the memory capacity of neural tissue. *Neuron* **29**, 779–96 (2001).
4. Poirazi, P., Brannon, T. & Mel, B.W. Arithmetic of subthreshold synaptic summation in a model CA1 pyramidal cell. *Neuron* **37**, 977–87 (2003).
5. Crochet, S., Poulet, J.F., Kremer, Y. & Petersen, C.C. Synaptic mechanisms underlying sparse coding of active touch. *Neuron* **69**, 1160–75 (2011).
6. Maass, W. & Bishop, C. *Pulsed Neural Networks* (MIT Press, 1998).
7. Gerstner, W. & Kistler, W. *Spiking Neuron Models* (Cambridge University Press, 2002).
8. Rieke, F., Warland, D., de Ruyter van Steveninck, R. & Bialek, W. *Spikes* (MIT Press, 1996).
9. Deneve, S. Bayesian spiking neurons I: inference. *Neural Comput.* **20**, 91–117 (2008).
10. Dayan, P. & Abbot, L.F. *Theoretical neuroscience* (The MIT press, 2001).
11. Pfister, J., Dayan, P. & Lengyel, M. Know thy neighbour: a normative theory of synaptic depression. *Adv. Neural Inf. Proc. Sys.* **22**, 1464–1472 (2009).
12. Pfister, J., Dayan, P. & Lengyel, M. Synapses with short-term plasticity are optimal estimators of presynaptic membrane potentials. *Nat. Neurosci.* **13**, 1271–1275 (2010).
13. Poulet, J.F. & Petersen, C.C. Internal brain state regulates membrane potential synchrony in barrel cortex of behaving mice. *Nature* **454**, 881–5 (2008).
14. Doucet, A., De Freitas, N. & Gordon, N. *Sequential Monte Carlo Methods in Practice* (Springer, New York, 2001).
15. Rall, W. Branching dendritic trees and motoneuron membrane resistivity. *Exp. Neurol.* **1**, 491–527 (1959).
16. Hoffman, D.A., Magee, J.C., Colbert, C.M. & Johnston, D. K+ channel regulation of signal propagation in dendrites of hippocampal pyramidal neurons. *Nature* **387**, 869–75 (1997).
17. Cash, S. & Yuste, R. Linear summation of excitatory inputs by CA1 pyramidal neurons. *Neuron* **22**, 383–94 (1999).
18. Gasparini, S., Migliore, M. & Magee, J.C. On the initiation and propagation of dendritic spikes in CA1 pyramidal neurons. *J. Neurosci.* **24**, 11046–56 (2004).
19. Polsky, A., Mel, B.W. & Schiller, J. Computational subunits in thin dendrites of pyramidal cells. *Nat. Neurosci.* **7**, 621–7 (2004).
20. Margulis, M. & Tang, C.M. Temporal integration can readily switch between sublinear and supralinear summation. *J. Neurophysiol.* **79**, 2809–13 (1998).
21. Hausser, M., Spruston, N. & Stuart, G.J. Diversity and dynamics of dendritic signaling. *Science* **290**, 739–44 (2000).
22. Poirazi, P., Brannon, T. & Mel, B.W. Pyramidal neuron as two-layer neural network. *Neuron* **37**, 989–99 (2003).
23. Huys, Q.J., Zemel, R.S., Natarajan, R. & Dayan, P. Fast population coding. *Neural Comput.* **19**, 404–41 (2007).
24. Natarajan, R., Huys, Q.J.M., Dayan, P. & Zemel, R.S. Encoding and decoding spikes for dynamics stimuli. *Neural Computation* **20**, 2325–2360 (2008).
25. Gerwinn, S., Macke, J. & Bethge, M. Bayesian population decoding with spiking neurons. *Frontiers in Computational Neuroscience* **3** (2009).
26. Losonczy, A. & Magee, J.C. Integrative properties of radial oblique dendrites in hippocampal CA1 pyramidal neurons. *Neuron* **50**, 291–307 (2006).
27. Bock, D.D. *et al.* Network anatomy and in vivo physiology of visual cortical neurons. *Nature* **471**, 177–82 (2011).
28. Ko, H. *et al.* Functional specificity of local synaptic connections in neocortical networks. *Nature* (2011).
29. Losonczy, A., Makara, J.K. & Magee, J.C. Compartmentalized dendritic plasticity and input feature storage in neurons. *Nature* **452**, 436–41 (2008).
30. Makara, J.K., Losonczy, A., Wen, Q. & Magee, J.C. Experience-dependent compartmentalized dendritic plasticity in rat hippocampal CA1 pyramidal neurons. *Nat. Neurosci.* **12**, 1485–7 (2009).
31. Butz, M., Worgotter, F. & van Ooyen, A. Activity-dependent structural plasticity. *Brain Res. Rev.* **60**, 287–305 (2009).

